# Online learning from finite training sets: An analytical case study

**Peter Sollich***
Department of Physics
University of Edinburgh
Edinburgh EH9 3JZ, U.K.
P.Sollich@ed.ac.uk

**David Barber**[†]
Neural Computing Research Group
Department of Applied Mathematics
Aston University
Birmingham B4 7ET, U.K.
D.Barber@aston.ac.uk

## Abstract

We analyse online learning from *finite* training sets at *non-infinitesimal* learning rates $\eta$. By an extension of statistical mechanics methods, we obtain exact results for the time-dependent generalization error of a linear network with a large number of weights $N$. We find, for example, that for small training sets of size $p \approx N$, larger learning rates can be used without compromising asymptotic generalization performance or convergence speed. Encouragingly, for optimal settings of $\eta$ (and, less importantly, weight decay $\lambda$) at given final learning time, the generalization performance of online learning is essentially as good as that of offline learning.

## 1  INTRODUCTION

The analysis of online (gradient descent) learning, which is one of the most common approaches to supervised learning found in the neural networks community, has recently been the focus of much attention [1]. The characteristic feature of online learning is that the weights of a network ('student') are updated each time a new training example is presented, such that the error on this example is reduced. In offline learning, on the other hand, the total error on all examples in the training set is accumulated before a gradient descent weight update is made. Online and

offline learning are equivalent only in the limiting case where the learning rate $\eta \to 0$ (see, *e.g.*, [2]). The main quantity of interest is normally the evolution of the generalization error: How well does the student approximate the input-output mapping ('teacher') underlying the training examples after a given number of weight updates?

Most analytical treatments of online learning assume either that the size of the training set is infinite, or that the learning rate $\eta$ is vanishingly small. Both of these restrictions are undesirable: In practice, most training sets are finite, and non-infinitesimal values of $\eta$ are needed to ensure that the learning process converges after a reasonable number of updates. General results have been derived for the difference between online and offline learning to first order in $\eta$, which apply to training sets of any size (see, *e.g.*, [2]). These results, however, do not directly address the question of generalization performance. The most explicit analysis of the time evolution of the generalization error for finite training sets was provided by Krogh and Hertz [3] for a scenario very similar to the one we consider below. Their $\eta \to 0$ (*i.e.*, offline) calculation will serve as a baseline for our work. For finite $\eta$, progress has been made in particular for so-called soft committee machine network architectures [4, 5], but only for the case of infinite training sets.

Our aim in this paper is to analyse a simple model system in order to assess how the combination of non-infinitesimal learning rates $\eta$ and finite training sets (containing $\alpha$ examples per weight) affects online learning. In particular, we will consider the dependence of the asymptotic generalization error on $\eta$ and $\alpha$, the effect of finite $\alpha$ on both the critical learning rate and the learning rate yielding optimal convergence speed, and optimal values of $\eta$ and weight decay $\lambda$. We also compare the performance of online and offline learning and discuss the extent to which infinite training set analyses are applicable for finite $\alpha$.

## 2 MODEL AND OUTLINE OF CALCULATION

We consider online training of a linear student network with input-output relation

$$y = \mathbf{w}^\mathrm{T}\mathbf{x}/\sqrt{N}.$$

Here $\mathbf{x}$ is an $N$-dimensional vector of real-valued inputs, $y$ the single real output and $\mathbf{w}$ the weight vector of the network. 'T' denotes the transpose of a vector and the factor $1/\sqrt{N}$ is introduced for convenience. Whenever a training example $(\mathbf{x}, y)$ is presented to the network, its weight vector is updated along the gradient of the squared error on this example, *i.e.*,

$$\Delta\mathbf{w} = -\eta\,\nabla_\mathbf{w}\tfrac{1}{2}(y - \mathbf{w}^\mathrm{T}\mathbf{x}/\sqrt{N})^2 = \eta\,(y\mathbf{x}/\sqrt{N} - \tfrac{1}{N}\mathbf{x}\mathbf{x}^\mathrm{T}\mathbf{w})$$

where $\eta$ is the learning rate. We are interested in online learning from finite training sets, where for each update an example is randomly chosen from a given set $\{(\mathbf{x}^\mu, y^\mu), \mu = 1\ldots p\}$ of $p$ training examples. (The case of cyclical presentation of examples [6] is left for future study.) If example $\mu$ is chosen for update $n$, the weight vector is changed to

$$\mathbf{w}_{n+1} = \{1 - \eta\,\tfrac{1}{N}[\mathbf{x}^\mu(\mathbf{x}^\mu)^\mathrm{T} + \gamma]\}\mathbf{w}_n + \eta\,y^\mu\mathbf{x}^\mu/\sqrt{N} \tag{1}$$

Here we have also included a weight decay $\gamma$. We will normally parameterize the strength of the weight decay in terms of $\lambda = \gamma\alpha$ (where $\alpha = p/N$ is the number

of examples per weight), which plays the same role as the weight decay commonly used in offline learning [3]. For simplicity, all student weights are assumed to be initially zero, *i.e.*, $\mathbf{w}_{n=0} = \mathbf{0}$.

The main quantity of interest is the evolution of the *generalization error* of the student. We assume that the training examples are generated by a linear 'teacher', *i.e.*, $y^\mu = \mathbf{w}_*{}^T \mathbf{x}^\mu / \sqrt{N} + \xi^\mu$, where $\xi^\mu$ is zero mean additive noise of variance $\sigma^2$. The teacher weight vector is taken to be normalized to $\mathbf{w}_*{}^2 = N$ for simplicity, and the input vectors are assumed to be sampled randomly from an isotropic distribution over the hypersphere $\mathbf{x}^2 = N$. The generalization error, defined as the average of the squared error between student and teacher outputs for random inputs, is then

$$\epsilon_g = \frac{1}{2N}(\mathbf{w}_n - \mathbf{w}_*)^2 = \frac{1}{2N}\mathbf{v}_n^2 \quad \text{where} \quad \mathbf{v}_n = \mathbf{w}_n - \mathbf{w}_*.$$

In order to make the scenario analytically tractable, we focus on the limit $N \to \infty$ of a large number of input components and weights, taken at constant number of examples per weight $\alpha = p/N$ and updates per weight ('learning time') $t = n/N$. In this limit, the generalization error $\epsilon_g(t)$ becomes self-averaging and can be calculated by averaging both over the random selection of examples from a given training set and over all training sets. Our results can be straightforwardly extended to the case of perceptron teachers with a nonlinear transfer function, as in [7].

The usual statistical mechanical approach to the online learning problem expresses the generalization error in terms of 'order parameters' like $R = \frac{1}{N}\mathbf{w}_n^T \mathbf{w}_*$ whose (self-averaging) time evolution is determined from appropriately averaged update equations. This method works because for *infinite* training sets, the average order parameter updates can again be expressed in terms of the order parameters alone. For *finite* training sets, on the other hand, the updates involve new order parameters such as $R_1 = \frac{1}{N}\mathbf{w}_n^T \mathbf{A}\mathbf{w}_*$, where $\mathbf{A}$ is the correlation matrix of the training inputs, $\mathbf{A} = \frac{1}{N}\sum_{\mu=1}^{p}\mathbf{x}^\mu(\mathbf{x}^\mu)^T$. *Their* time evolution is in turn determined by order parameters involving higher powers of $\mathbf{A}$, yielding an infinite hierarchy of order parameters. We solve this problem by considering instead *order parameter* (generating) *functions* [8] such as a generalized form of the generalization error $\epsilon(t; h) = \frac{1}{2N}\mathbf{v}_n^T \exp(h\mathbf{A})\mathbf{v}_n$. This allows powers of $\mathbf{A}$ to be obtained by differentiation with respect to $h$, resulting in a closed system of (partial differential) equations for $\epsilon(t; h)$ and $R(t; h) = \frac{1}{N}\mathbf{w}_n^T \exp(h\mathbf{A})\mathbf{w}_*$.

The resulting equations and details of their solution will be given in a future publication. The final solution is most easily expressed in terms of the Laplace transform of the generalization error

$$\hat{\epsilon}_g(z) = \frac{\eta}{\alpha}\int_0^\infty dt\, \epsilon_g(t)e^{-z(\eta/\alpha)t} = \frac{\epsilon_1(z) + \eta\epsilon_2(z) + \eta^2\epsilon_3(z)}{1 - \eta\epsilon_4(z)} \tag{2}$$

The functions $\epsilon_i(z)$ $(i = 1 \ldots 4)$ can be expressed in closed form in terms of $\alpha$, $\sigma^2$ and $\lambda$ (and, of course, $z$). The Laplace transform (2) yields directly the asymptotic value of the generalization error, $\epsilon_\infty = \epsilon_g(t \to \infty) = \lim_{z\to 0} z\hat{\epsilon}_g(z)$, which can be calculated analytically. For finite learning times $t$, $\epsilon_g(t)$ is obtained by numerical inversion of the Laplace transform.

## 3   RESULTS AND DISCUSSION

We now discuss the consequences of our main result (2), focusing first on the asymptotic generalization error $\epsilon_\infty$, then the convergence speed for large learning times,

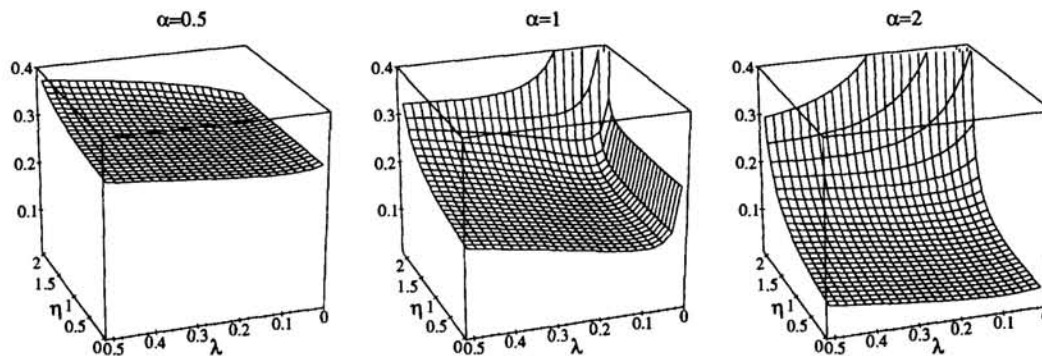

Figure 1: Asymptotic generalization error $\epsilon_\infty$ vs $\eta$ and $\lambda$. $\alpha$ as shown, $\sigma^2 = 0.1$.

and finally the behaviour at small $t$. For numerical evaluations, we generally take $\sigma^2 = 0.1$, corresponding to a sizable noise-to-signal ratio of $\sqrt{0.1} \approx 0.32$.

The asymptotic generalization error $\epsilon_\infty$ is shown in Fig. 1 as a function of $\eta$ and $\lambda$ for $\alpha = 0.5, 1, 2$. We observe that it is minimal for $\lambda = \sigma^2$ and $\eta = 0$, as expected from corresponding results for offline learning [3][1]. We also read off that for fixed $\lambda$, $\epsilon_\infty$ is an increasing function of $\eta$: The larger $\eta$, the more the weight updates tend to overshoot the minimum of the (total, *i.e.*, offline) training error. This causes a diffusive motion of the weights around their average asymptotic values [2] which increases $\epsilon_\infty$. In the absence of weight decay ($\lambda = 0$) and for $\alpha < 1$, however, $\epsilon_\infty$ is independent of $\eta$. In this case the training data can be fitted perfectly; every term in the total sum-of-squares training error is then zero and online learning does not lead to weight diffusion because all individual updates vanish. In general, the relative increase $\epsilon_\infty(\eta)/\epsilon_\infty(\eta = 0) - 1$ due to nonzero $\eta$ depends significantly on $\alpha$. For $\eta = 1$ and $\alpha = 0.5$, for example, this increase is smaller than 6% for all $\lambda$ (at $\sigma^2 = 0.1$), and for $\alpha = 1$ it is at most 13%. This means that in cases where training data is limited ($p \approx N$), $\eta$ can be chosen fairly large in order to optimize learning speed, without seriously affecting the asymptotic generalization error. In the large $\alpha$ limit, on the other hand, one finds $\epsilon_\infty = (\sigma^2/2)[1/\alpha + \eta/(2 - \eta)]$. The relative increase over the value at $\eta = 0$ therefore grows linearly with $\alpha$; already for $\alpha = 2$, increases of around 50% can occur for $\eta = 1$.

Fig. 1 also shows that $\epsilon_\infty$ diverges as $\eta$ approaches a critical learning rate $\eta_c$: As $\eta \to \eta_c$, the 'overshoot' of the weight update steps becomes so large that the weights eventually diverge. From the Laplace transform (2), one finds that $\eta_c$ is determined by $\eta_c \epsilon_4(z = 0) = 1$; it is a function of $\alpha$ and $\lambda$ only. As shown in Fig. 2b-d, $\eta_c$ increases with $\lambda$. This is reasonable, as the weight decay reduces the length of the weight vector at each update, counteracting potential weight divergences. In the small and large $\alpha$ limit, one has $\eta_c = 2(1 + \lambda)$ and $\eta_c = 2(1 + \lambda/\alpha)$, respectively. For constant $\lambda$, $\eta_c$ therefore decreases[2] with $\alpha$ (Fig. 2b-d).

We now turn to the large $t$ behaviour of the generalization error $\epsilon_g(t)$. For small $\eta$, the most slowly decaying contribution (or 'mode') to $\epsilon_g(t)$ varies as $\exp(-ct)$, its

decay constant $c = \eta[\lambda + (\sqrt{\alpha} - 1)^2]/\alpha$ scaling linearly with $\eta$, the size of the weight updates, as expected (Fig. 2a). For small $\alpha$, the condition $ct \gg 1$ for $\epsilon_g(t)$ to have reached its asymptotic value $\epsilon_\infty$ is $\eta(1 + \lambda)(t/\alpha) \gg 1$ and scales with $t/\alpha$, which is the number of times each training example has been used. For large $\alpha$, on the other hand, the condition becomes $\eta t \gg 1$: The size of the training set drops out since convergence occurs before repetitions of training examples become significant.

For larger $\eta$, the picture changes due to a new 'slow mode' (arising from the denominator of (2)). Interestingly, this mode exists only for $\eta$ above a finite threshold $\eta_{\min} = 2/(\alpha^{1/2} + \alpha^{-1/2} - 1)$. For finite $\alpha$, it could therefore not have been predicted from a small $\eta$ expansion of $\epsilon_g(t)$. Its decay constant $c_{\text{slow}}$ decreases to zero as $\eta \to \eta_c$, and crosses that of the normal mode at $\eta_x(\alpha, \lambda)$ (Fig. 2a). For $\eta > \eta_x$, the slow mode therefore determines the convergence speed for large $t$, and fastest convergence is obtained for $\eta = \eta_x$. However, it may still be advantageous to use lower values of $\eta$ in order to lower the asymptotic generalization error (see below); values of $\eta > \eta_x$ would deteriorate both convergence speed and asymptotic performance. Fig. 2b-d shows the dependence of $\eta_{\min}$, $\eta_x$ and $\eta_c$ on $\alpha$ and $\lambda$. For $\lambda$ not too large, $\eta_x$ has a maximum at $\alpha \approx 1$ (where $\eta_x \approx \eta_c$), while decaying as $\eta_x = 1 + 2\alpha^{-1/2} \approx \frac{1}{2}\eta_c$ for larger $\alpha$. This is because for $\alpha \approx 1$ the (total training) error surface is very anisotropic around its minimum in weight space [9]. The steepest directions determine $\eta_c$ and convergence along them would be fastest for $\eta = \frac{1}{2}\eta_c$ (as in the isotropic case). However, the overall convergence speed is determined by the shallow directions, which require maximal $\eta \approx \eta_c$ for fastest convergence.

Consider now the small $t$ behaviour of $\epsilon_g(t)$. Fig. 3 illustrates the dependence of $\epsilon_g(t)$ on $\eta$; comparison with simulation results for $N = 50$ clearly confirms our calculations and demonstrates that finite $N$ effects are not significant even for such fairly small $N$. For $\alpha = 0.7$ (Fig. 3a), we see that nonzero $\eta$ acts as effective update noise, eliminating the minimum in $\epsilon_g(t)$ which corresponds to over-training [3]. $\epsilon_\infty$ is also seen to be essentially independent of $\eta$ as predicted for the small value of $\lambda = 10^{-4}$ chosen. For $\alpha = 5$, Fig. 3b clearly shows the increase of $\epsilon_\infty$ with $\eta$. It also illustrates how convergence first speeds up as $\eta$ is increased from zero and then slows down again as $\eta_c \approx 2$ is approached.

Above, we discussed optimal settings of $\eta$ and $\lambda$ for minimal asymptotic generalization error $\epsilon_\infty$. Fig. 4 shows what happens if we minimize $\epsilon_g(t)$ instead for a given *final learning time* $t$, corresponding to a fixed amount of computational effort for training the network. As $t$ increases, the optimal $\eta$ decreases towards zero as required by the tradeoff between asymptotic performance and convergence

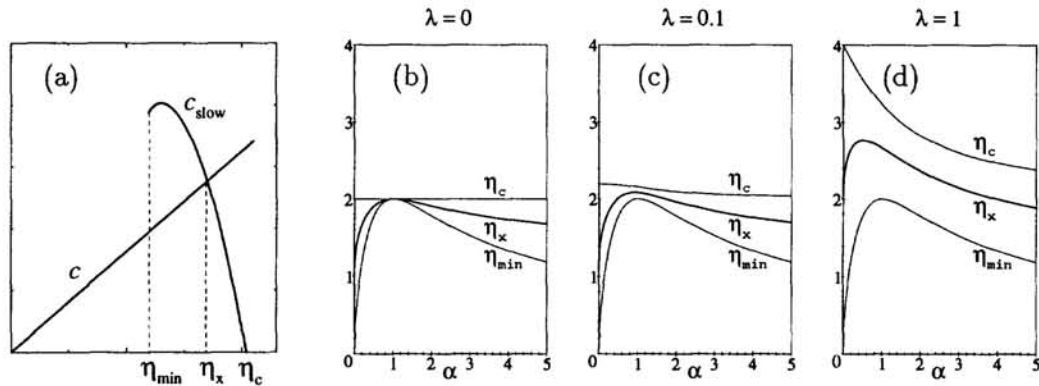

Figure 2: Definitions of $\eta_{\min}$, $\eta_x$ and $\eta_c$, and their dependence on $\alpha$ (for $\lambda$ as shown).

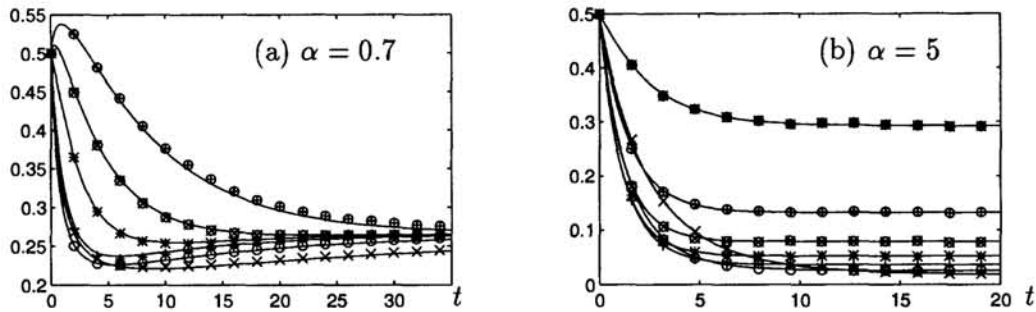

Figure 3: $\epsilon_g$ vs $t$ for different $\eta$. Simulations for $N = 50$ are shown by symbols (standard errors less than symbol sizes). $\lambda=10^{-4}$, $\sigma^2=0.1$, $\alpha$ as shown. The learning rate $\eta$ increases from below (at large $t$) over the range (a) $0.5\ldots1.95$, (b) $0.5\ldots1.75$.

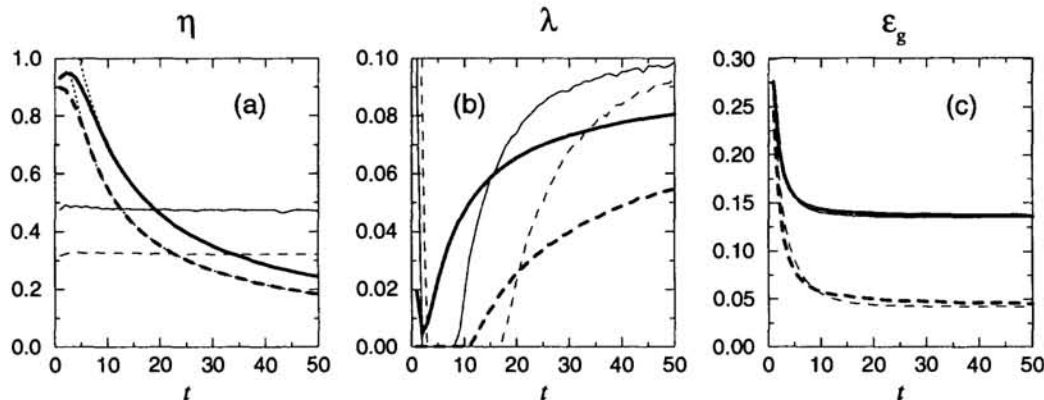

Figure 4: Optimal $\eta$ and $\lambda$ vs given final learning time $t$, and resulting $\epsilon_g$. Solid/dashed lines: $\alpha=1$ / $\alpha=2$; bold/thin lines: online/offline learning. $\sigma^2=0.1$. Dotted lines in (a): Fits of form $\eta = (a+b\ln t)/t$ to optimal $\eta$ for online learning.

speed. Minimizing $\epsilon_g(t) \approx \epsilon_\infty + \text{const} \cdot \exp(-ct) \approx c_1 + \eta c_2 + c_3 \exp(-c_4\eta t)$ leads to $\eta_{\text{opt}} = (a + b\ln t)/t$ (with some constants $a$, $b$, $c_{1\ldots4}$). Although derived for small $\eta$, this functional form (dotted lines in Fig. 4a) also provides a good description down to fairly small $t$, where $\eta_{\text{opt}}$ becomes large. The optimal weight decay $\lambda$ increases[3] with $t$ towards the limiting value $\sigma^2$. However, optimizing $\lambda$ is much less important than choosing the right $\eta$: Minimizing $\epsilon_g(t)$ for fixed $\lambda$ yields almost the same generalization error as optimizing both $\eta$ and $\lambda$ (we omit detailed results here[4]). It is encouraging to see from Fig. 4c that after as few as $t = 10$ updates per weight with optimal $\eta$, the generalization error is almost indistinguishable from its optimal value for $t \to \infty$ (this also holds if $\lambda$ is kept fixed). Optimization of the learning rate should therefore be worthwhile in most practical scenarios.

In Fig. 4c, we also compare the performance of online learning to that of offline learning (calculated from the appropriate discrete time version of [3]), again with

optimized values of $\eta$ and $\lambda$ for given $t$. The performance loss from using online instead of offline learning is seen to be negligible. This may seem surprising given the effective noise on weight updates implied by online learning, in particular for small $t$. However, comparing the respective optimal learning rates (Fig. 4a), we see that online learning makes up for this deficiency by allowing larger values of $\eta$ to be used (for large $\alpha$, for example, $\eta_c(\text{offline}) = 2/\alpha \ll \eta_c(\text{online}) = 2$).

Finally, we compare our finite $\alpha$ results with those for the limiting case $\alpha \to \infty$. Good agreement exists for any learning time $t$ if the asymptotic generalization error $\epsilon_\infty(\alpha < \infty)$ is dominated by the contribution from the nonzero learning rate $\eta$ (as is the case for $\alpha \to \infty$). In practice, however, one wants $\eta$ to be small enough to make only a negligible contribution to $\epsilon_\infty(\alpha < \infty)$; in this regime, the $\alpha \to \infty$ results are essentially useless.

# 4   CONCLUSIONS

The main theoretical contribution of this paper is the extension of the statistical mechanics method of order parameter dynamics to the dynamics of order parameter (generating) functions. The results that we have obtained for a simple linear model system are also of practical relevance. For example, the calculated dependence on $\eta$ of the asymptotic generalization error $\epsilon_\infty$ and the convergence speed shows that, in general, sizable values of $\eta$ can be used for training sets of limited size ($\alpha \approx 1$), while for larger $\alpha$ it is important to keep learning rates small. We also found a simple functional form for the dependence of the optimal $\eta$ on a given final learning time $t$. This could be used, for example, to estimate the optimal $\eta$ for large $t$ from test runs with only a small number of weight updates. Finally, we found that for optimized $\eta$ online learning performs essentially as well as offline learning, whether or not the weight decay $\lambda$ is optimized as well. This is encouraging, since online learning effectively induces noisy weight updates. This allows it to cope better than offline learning with the problem of local (training error) minima in realistic neural networks. Online learning has the further advantage that the critical learning rates are not significantly lowered by input distributions with nonzero mean, whereas for offline learning they are significantly reduced [10]. In the future, we hope to extend our approach to dynamic ($t$-dependent) optimization of $\eta$ (although performance improvements over optimal fixed $\eta$ may be small [6]), and to more complicated network architectures in which the crucial question of local minima can be addressed.

## Footnotes

* Royal Society Dorothy Hodgkin Research Fellow

† Supported by EPSRC grant GR/J75425: Novel Developments in Learning Theory for Neural Networks

[1]The optimal value of the *unscaled* weight decay decreases with $\alpha$ as $\gamma = \sigma^2/\alpha$, because for large training sets there is less need to counteract noise in the training data by using a large weight decay.

[2]Conversely, for constant $\gamma$, $\eta_c$ *increases* with $\alpha$ from $2(1+\gamma\alpha)$ to $2(1+\gamma)$: For large $\alpha$, the weight decay is applied more often between repeat presentations of a training example that would otherwise cause the weights to diverge.

[3]One might have expected the opposite effect of having larger $\lambda$ at low $t$ in order to 'contain' potential divergences from the larger optimal learning rates $\eta$. However, smaller $\lambda$ tends to make the asymptotic value $\epsilon_\infty$ less sensitive to large values of $\eta$ as we saw above, and we conclude that this effect dominates.

[4]For fixed $\lambda < \sigma^2$, where $\epsilon_g(t)$ has an over-training minimum (see Fig. 3a), the asymptotic behaviour of $\eta_{\text{opt}}$ changes to $\eta_{\text{opt}} \propto t^{-1}$ (without the $\ln t$ factor), corresponding to a fixed effective learning time $\eta t$ required to reach this minimum.

# References

[1] See for example: *The dynamics of online learning*. Workshop at *NIPS'95*.
[2] T. Heskes and B. Kappen. *Phys. Rev. A*, 44:2718, 1991.
[3] A. Krogh and J. A. Hertz. *J. Phys. A*, 25:1135, 1992.
[4] D. Saad and S. Solla. *Phys. Rev. E*, 52:4225, 1995; also in *NIPS-8*.
[5] M. Biehl and H. Schwarze. *J. Phys. A*, 28:643–656, 1995.
[6] Z.-Q. Luo. *Neur. Comp.*, 3:226, 1991; T. Heskes and W. Wiegerinck. *IEEE Trans. Neur. Netw.*, 7:919, 1996.
[7] P. Sollich. *J. Phys. A*, 28:6125, 1995.
[8] L. L. Bonilla, F. G. Padilla, G. Parisi and F. Ritort. *Europhys. Lett.*, 34:159, 1996; *Phys. Rev. B*, 54:4170, 1996.
[9] J. A. Hertz, A. Krogh and G. I. Thorbergsson. *J. Phys. A*, 22:2133, 1989.
[10] T. L. H. Watkin, A. Rau and M. Biehl. *Rev. Modern Phys.*, 65:499, 1993.